# A B-P ANN Commodity Trader

Joseph E. Collard
Martingale Research Corporation
100 Allentown Pkwy., Suite 211
Allen, Texas 75002

## Abstract

An Artificial Neural Network (ANN) is trained to recognize a buy/sell (long/short) pattern for a particular commodity future contract. The Back-Propagation of errors algorithm was used to encode the relationship between the Long/Short desired output and 18 fundamental variables plus 6 (or 18) technical variables into the ANN. Trained on one year of past data the ANN is able to predict long/short market positions for 9 months in the future that would have made $10,301 profit on an investment of less than $1000.

## 1 INTRODUCTION

An Artificial Neural Network (ANN) is trained to recognize a long/short pattern for a particular commodity future contract. The Back-Propagation of errors algorithm was used to encode the relationship between the Long/Short desired output and 18 fundamental variables plus 6 (or 18) technical variables into the ANN.

## 2 NETWORK ARCHITECTURE

The ANNs used were simple, feed forward, single hidden layer networks with no input units, N hidden units and one output unit. See Figure 1. N varied from six (6) through sixteen (16) hidden units.

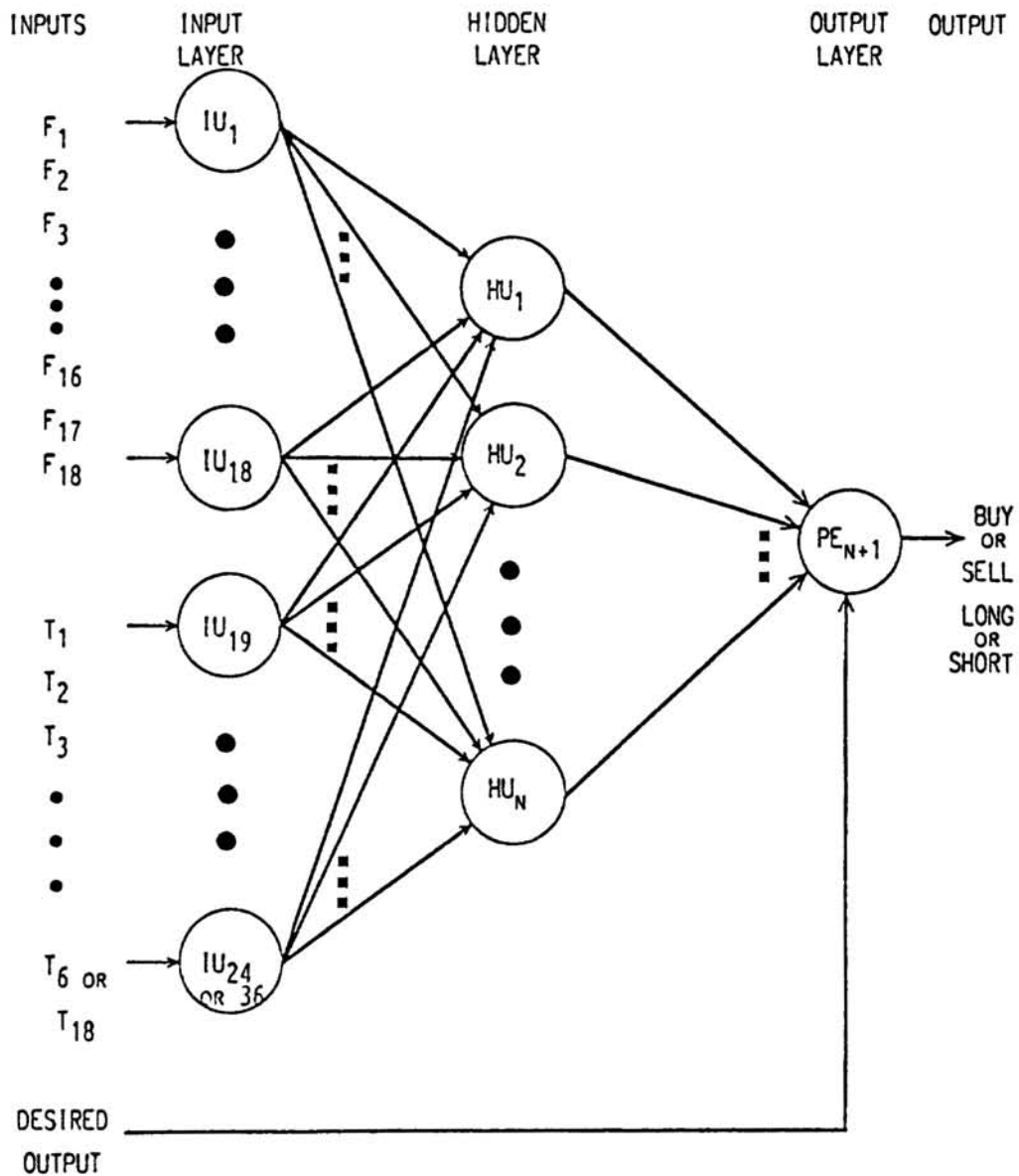

Figure 1.   The Network Architecture

## 3    TRAINING PROCEDURE

Back Propagation of Errors Algorithm:    The ANN was trained using the well known ANN training algorithm called Back Propagation of Errors which will not be elaborated on here.

A Few Mods to the Algorithm:    We are using the algorithm above with three changes.  The changes, when implemented and tested on the standard exclusive-or problem, resulted in a trained, one hidden unit network after 60-70 passes through the 4 pattern vectors.   This compares  to the 245 passes cited by Rumelhart [2].  Even with a 32 hidden unit network, Yves found the average number of passes to be 120 [2].

The modifications to standard back propagation are:

1. Minimum Slope Term in the Derivative of the Activation Function [John Denker at Bell Labs].
2. Using the Optional Momentum Term [2].
3. Weight change frequency [1].

## 4    DATA

In all cases, the six market technical variables (Open, High, Low, Close, Open Interest and Volume) were that trading day's data for the "front month" commodity contract (roughly speaking, the most active month's commodity contract).

The first set of training data consisted of 105 or 143 "high confidence" trading days in 1988. Each trading day had associated with it a twenty-five component pattern vector (25-vector) consisting of eighteen fundamental variables, such as weather indicators and seasonal indicators, plus the six market technical variables for the trading day, and finally, the EXPERT's hindsight long/short position. The test data for these networks was all 253 25-vectors in 1988.

The next training data set consisted of all 253 trading days in 1988. Again each trading day had associated with it a 25-vector consisting of the same eighteen fundamental variables plus the six market technical variables and finally, the EXPERT's long/short position. The test set for these networks consisted of 25-vectors from the first 205 trading days in 1989.

Finally, the last set of training data consisted of the last 251 trading days in 1988. For this set each trading day had associated with it a 37 component pattern vector (37-vector) consisting of the same eighteen fundamental variables plus six market technical variables for that trading day, six market technical variables for the previous trading day, six market technical variables for the two days previous trading day, and finally, the EXPERT's long/short position. The test set for these networks consisted of 37-vectors from the first 205 trading days in 1989.

## 5    RESULTS

The results for 7 trained networks are summarized in Table 1.

Table 1.  Study Results

| # | SIZE/IN | TRAIN. | % @ ε-XPT | PROFIT/RTS | TEST | PROFIT/RTS |
|---|---------|--------|-----------|------------|------|------------|
| 005 | 6-1 /24 | 105-'88 | 100 @.125 | | 253-'88 | 76% |
| 006 | 6-1 /24 | 143-'88 | 99 @.125-1 | | 253-'88 | 82% |
| Targets >>> | | 253-'88 | ---------- | $25,296/10 | 205-'89 | $14,596/ 6 |
| 009 | 10-1/24 | 253-'88 | 98 @ .25-4 | $24,173/14 | 205-'89 | $ 7,272/ 6 |
| 010 | 6-1 /24 | 105-'88 | 100 @ .1 | $17,534/13 | 253-'88 | 80% |
| Targets >>> | | 251-'88 | --------- | $24,819/10 | 205-'89 | $14,596/ 6 |
| 011 | 10-1/36 | 251-'88 | 98 @ .25-4 | $23,370/14 | 205-'89 | $ 7,272/ 6 |
| 012 | 13-1/36 | 251-'88 | 97 @ .25-7 | $22,965/12 | 205-'89 | $ 6,554/14 |
| 013 | 16-1/36 | 251-'88 | 99 @ .25-3 | $22,495/12 | 205-'89 | $10,301/19 |

The column headings for Table 1 have the following meanings:

| | |
|---|---|
| # | The numerical designation of the Network. |
| Size/In | The hidden-output layer dimensions and the number of inputs to the network. |
| Train. | The number of days and year of the training set. |
| % @ ε-Xpt | The percent of the training data encoded in the network at less than ε error - the number of days not encoded. |
| Profit RTs | The profit computed for the training or test set and how many round turns (RTs) it required for that profit.  Or, if the profit calculation was not yet available, then the percent the network is in agreement with the EXPERT. |
| Test set | The number of trading days/year of the test set. |

Figure 2 shows how well the 013 network agrees with its training set's long/short positions.  The NET 19 INPUT curve is the commodities price curve for 1988's 251 trading days.

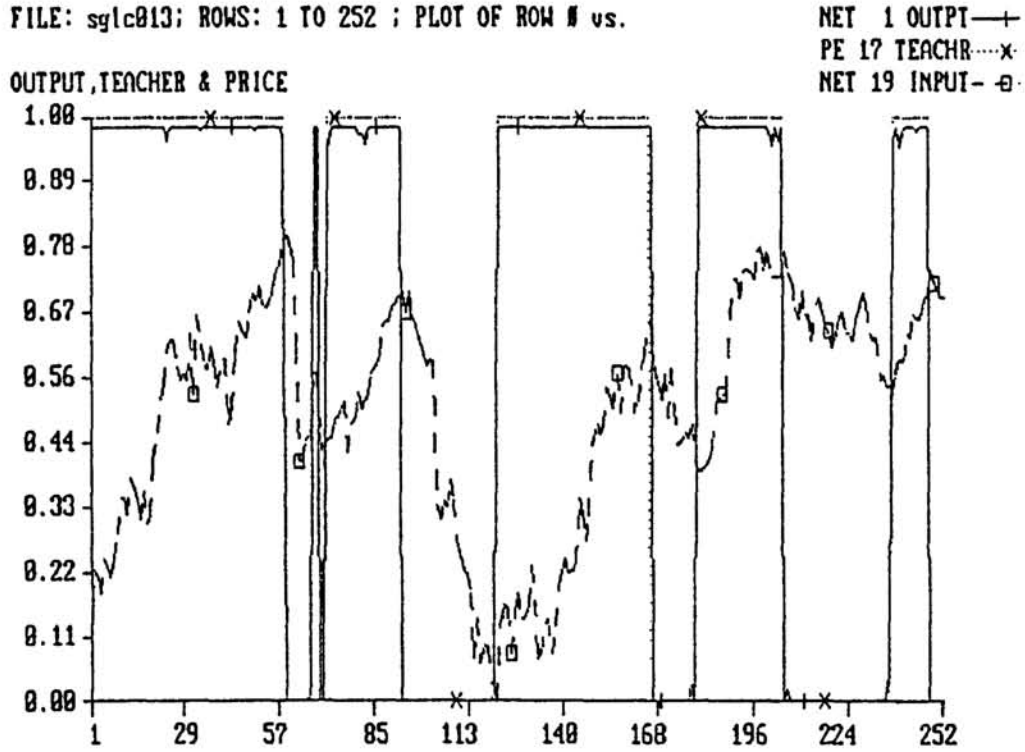

Figure 2.    Trained Network 013

Figure 3 shows the corresponding profit plot on the training data for both the EXPERT and network 013.

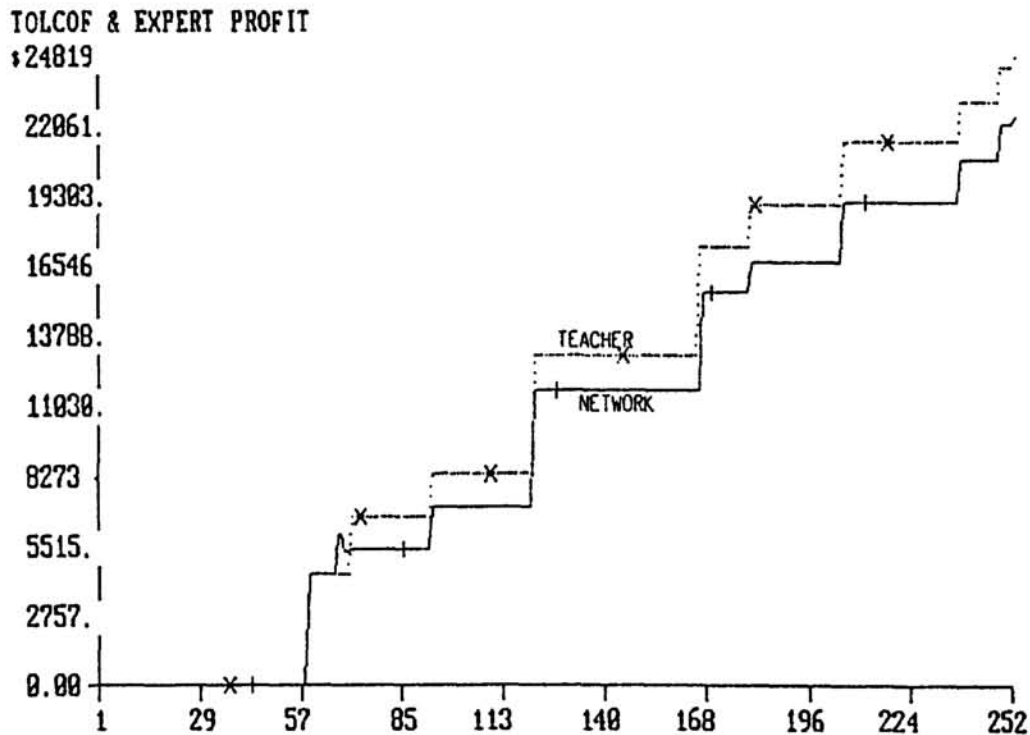

Figure 3.    Network 013's and EXPERT's Profit for 1988 data

Figure 4 is the profit plot for the network when tested on the first 205 trading days in 1989. Two significant features should be noted in Figure 4. The network's profit step function is almost always increasing. The profit is never negative.

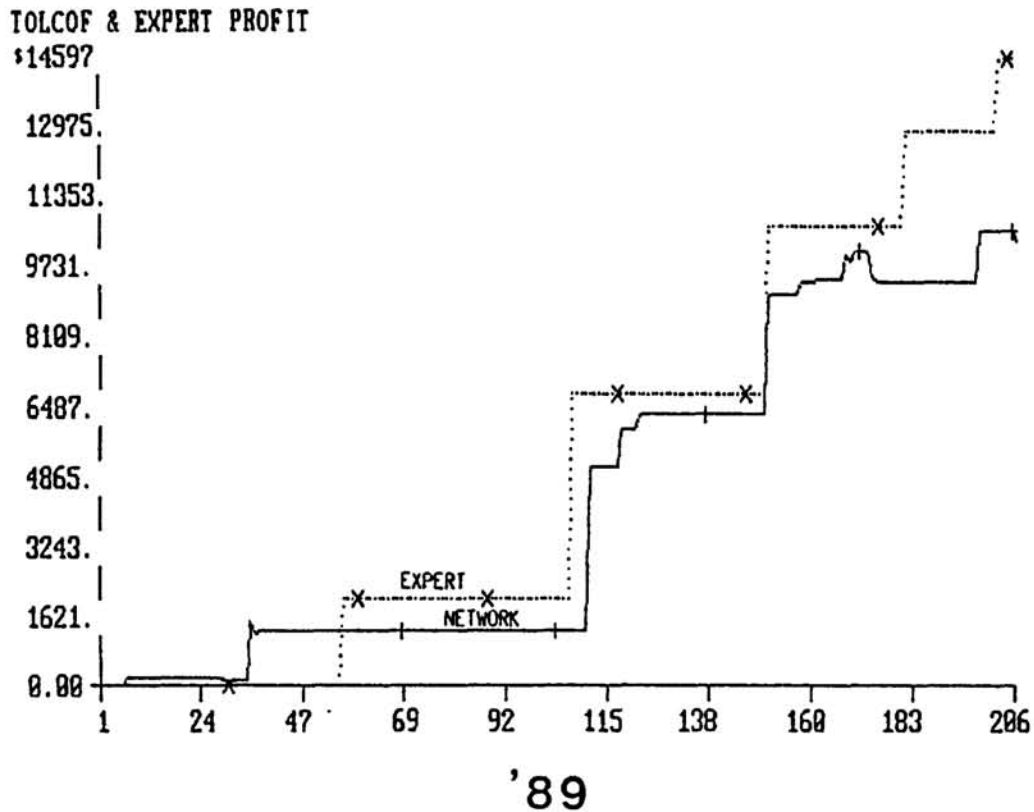

Figure 4.    Network 013's and EXPERT's Profit for 1989 data

## 6    REFERENCES

1. Pao, Y.- H., Adaptive Pattern Recognition and Neural Networks, Addison Wesley, 1989.
2. Rumelhart, D., and McClelland, J., Parallel Distributed Processing, MIT Press, 1986.
